# Towards social robots: Automatic evaluation of human-robot interaction by face detection and expression classification

**M.S. Bartlett[1], G. Littlewort[1], I. Fasel[1,2], J. Chenu[1,2], T. Kanda[1,2],**
**H. Ishiguro[1,2], and J.R. Movellan[1,2]**
[1]Institute for Neural Computation, University of California, San Diego
[2]Intelligent Robotics and Communication Laboratory, ATR, Kyoto Japan.
Email: gwen, marni, ian, joel, javier @inc.ucsd.edu

## Abstract

Computer animated agents and robots bring a social dimension to human computer interaction and force us to think in new ways about how computers could be used in daily life. Face to face communication is a real-time process operating at a time scale of less than a second. In this paper we present progress on a perceptual primitive to automatically detect frontal faces in the video stream and code them with respect to 7 dimensions in real time: neutral, anger, disgust, fear, joy, sadness, surprise. The face finder employs a cascade of feature detectors trained with boosting techniques [13, 2]. The expression recognizer employs a novel combination of Adaboost and SVM's. The generalization performance to new subjects for a 7-way forced choice was 93.3% and 97% correct on two publicly available datasets. The outputs of the classifier change smoothly as a function of time, providing a potentially valuable representation to code facial expression dynamics in a fully automatic and unobtrusive manner. The system was deployed and evaluated for measuring spontaneous facial expressions in the field in an application for automatic assessment of human-robot interaction.

## 1 Introduction

Computer animated agents and robots bring a social dimension to human computer interaction and force us to think in new ways about how computers could be used in daily life. Face to face communication is a real-time process operating at a time scale of less than a second. Thus fulfilling the idea of machines that interact face to face with us requires development of robust real-time perceptive primitives. In this paper we present first steps towards the development of one such primitive: a system that automatically finds faces in the visual video stream and codes facial expression dynamics in real time. The system automatically detects frontal faces and codes them with respect to 7 dimensions: Joy, sadness, surprise, anger, disgust, fear, and neutral. Speed and accuracy are enhanced by a novel technique that combines feature selection based on Adaboost with feature integration based on support vector machines. We host an online demo of the system at http://mplab.ucsd.edu.

The system was trained and tested on two publicly avaliable datasets of facial expressions collected by experimental psychologists expert in facial behavior. In addition, we deployed and evaluated the system in an application for recognizing spontaneous facial expressions from continuous video in the field. We assess the system as a method for automatic measurement of human-robot interaction.

## 2 Face detection

We developed a real-time face-detection system based on [13] capable of detection and false positive rates equivalent to the best published results [11, 12, 10, 13]. The system consists of a cascade of classifiers trained by boosting techniques. Each classifier employs integral image filters reminiscent of Haar Basis functions, which can be computed very fast at any location and scale in constant time (see Figure 1). In a $24 \times 24$ pixel window, there are over 160,000 possible filters of this type. For each stage in the cascade, a subset of features are chosen using a feature selection procedure based on Adaboost [3].

We enhance the approach in [13] in the following ways: (1) Once a feature is selected by boosting, we refine the selection by finding the best performing single-feature classifier from a new set of filters generated by shifting and scaling the chosen filter by two pixels in each direction, as well as composite filters made by reflecting each shifted and scaled feature horizontally about the center and superimposing it on the original. This can be thought of as a single generation genetic algorithm, and is much faster than exhaustively searching for the best classifier among all 160,000 possible filters and their reflection-based cousins.

(2) While [13] use Adaboost in their feature selection algorithm, which requires binary classifiers, we employed Gentleboost, described in [4], which uses real valued features. Figure 2 shows the first two filters chosen by the system along with the real valued output of the weak learners (or tuning curves) built on those filters. Note the bimodal distribution of filter 2.

(3) We have also developed a training procedure so that after each single feature, the system can decide whether to test another feature or to make a decision. This system retains information about the continuous outputs of each feature detector rather than converting to binary decisions at each stage of the cascade. Preliminary results show potential for dramatic improvements in speed with no loss of accuracy over the current system.

The face detector was trained on 5000 faces and millions of non-face patches from about 8000 images collected from the web by Compaq Research Laboratories. Accuracy on the CMU-MIT dataset (a standard, public data set for benchmarking frontal face detection systems) is comparable to [13]. Because the strong classifiers early in the sequence need very few features to achieve good performance (the first stage can reject 60% of the non-faces using only 2 features, using only 20 simple operations, or about 60 microprocessor instructions), the average number of features that need to be evaluated for each window is very small, making the overall system very fast. The source code for the face detector is freely available at http://www.sourceforge.net/projects/kolmogorov.

## 3 Facial Expression Classification

### 3.1 Data set

The facial expression system was trained and tested on Cohn and Kanade's DFAT-504 dataset [6]. This dataset consists of 100 university students ranging in age from 18 to 30 years. 65% were female, 15% were African-American, and 3% were Asian or Latino. Videos were recoded in analog S-video using a camera located directly in front of the subject. Subjects were instructed by an experimenter to perform a series of 23 facial expres-

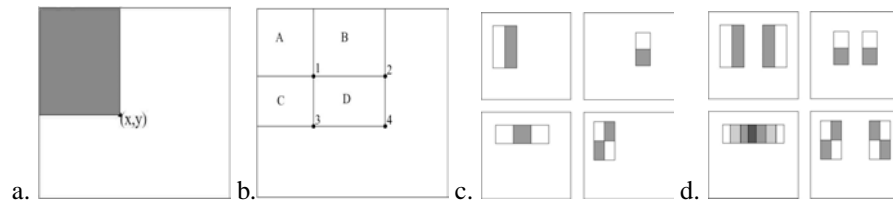

Figure 1: Integral image filters (after Viola & Jones, 2001 [13]). a. The value of the pixel at $(x, y)$ is the sum of all the pixels above and to the left. b. The sum of the pixels within rectangle $D$ can be computed as $4 + 1 - (2 + 3)$. (c) Each feature is computed by taking the difference of the sums of the pixels in the white boxes and grey boxes. Features include those shown in (c), as in [13], plus (d) the same features superimposed on their reflection about the Y axis.

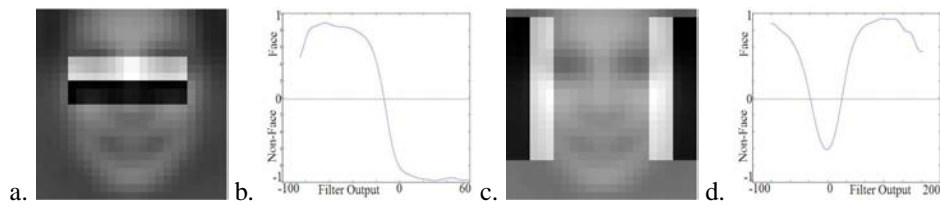

Figure 2: The first two features (a,c) and their respective tuning curves (b,d). Each feature is shown over the average face. The first tuning curve shows that a dark horizontal region over a bright horizontal region in the center of the window is evidence for a face, and for non-face otherwise. The output of the second filter is bimodal. Both a strong positive and a strong negative output is evidence for a face, while output closer to zero is evidence for non-face.

sions. Subjects began and ended each display with a neutral face. Before performing each display, an experimenter described and modeled the desired display. Image sequences from neutral to target display were digitized into 640 by 480 pixel arrays with 8-bit precision for grayscale values.

For our study, we selected 313 sequences from the dataset. The only selection criterion was that a sequence be labeled as one of the 6 basic emotions. The sequences came from 90 subjects, with 1 to 6 emotions per subject. The first and last frames (neutral and peak) were used as training images and for testing generalization to new subjects, for a total of 625 examples. The trained classifiers were later applied to the entire sequence.

All faces in this dataset were successfully detected. The automatically located faces were rescaled to 48x48 pixels.The typical distance between the centers of the eyes was roughly 24 pixels. A comparison was also made at double resolution (96x96). No further registration was performed. Other approaches to automatic facial expression recognition include explicit detection and alignment of internal facial features. The recognition system presented here performs well without that step, providing a considerable savings in processing time. The images were converted into a Gabor magnitude representation, using a bank of Gabor filters at 8 orientations and 5 spatial frequencies (4:16 pixels per cycle at 1/2 octave steps) [7].

## 4 SVM's and Adaboost

SVM performance was compared to Adaboost for emotion classification. The system performed a 7-way forced choice between the following emotion categories: Happiness, sadness, surprise, disgust, fear, anger, neutral. The classification was performed in two stages. First, seven binary classifiers were trained to discriminate each emotion from everything else. The emotion category decision was then implemented by choosing the classifier with the maximum output for the test example.

Support vector machines (SVM's) are well suited to this task because the high dimensionality of the Gabor representation does not affect training time for kernel classifiers. Linear, polynomial, and RBF kernels with Laplacian, and Gaussian basis functions were explored. Linear and RBF kernels employing a unit-width Gaussian performed best, and are presented here. Generalization to novel subjects was tested using leave-one-subject-out cross-validation. Results are presented in Table 1.

The features employed for the Adaboost emotion classifier were the individual Gabor filters. There were 48x48x40 = 92160 possible features. A subset of these filters was chosen using Adaboost. On each training round, the threshold and scale parameter of each filter was optimized and the feature that provided best performance on the boosted distribution was chosen.

During Adaboost, training for each emotion classifier continued until the distributions for the positive and negative samples were separated by a gap proportional to the widths of the two distributions. The total number of filters selected using this procedure was 538. Since Adaboost is significantly slower to train than SVM's, we did not do 'leave one subject out' cross validation. Instead we separated the subjects randomly into ten groups of roughly equal size and did 'leave one group out' cross validation. SVM performance for this training strategy is shown for comparison.

Results are shown in Table 1. The generalization performance, 85.0%, was comparable to linear SVM performance on the leave-group-out testing paradigm, but Adaboost was substantially faster, as shown in Table 2. Here, the system calculated the output of Gabor filters less efficiently, as the convolutions were done in pixel space rather than Fourier space, but the use of 200 times fewer Gabor filters nevertheless resulted in a substantial speed benefit.

## 5 AdaSVM's

Adaboost provides an added value of choosing which features are most informative to test at each step in the cascade. Figure 3a illustrates the first 5 Gabor features chosen for each emotion. The chosen features show no preference for direction, but the highest frequencies are chosen more often. Figure 3b shows the number of chosen features at each of the 5 wavelengths used.

A combination approach, in which the Gabor Features chosen by Adaboost were used as a reduced representation for training SVM's (AdaSVM's) outperformed Adaboost by 3.8 percent points, a difference that was statistically significant (z=1.99, p=0.02). AdaSVM's outperformed SVM's by an average of 2.7 percent points, an improvement that was marginally significant (z = 1.55, p = 0.06).

After examination of the frequency distribution of the Gabor filter selected by Adaboost, it became apparent that higher spatial frequency Gabors and higher resolution images could potentially improve performance. Indeed, by doubling the resolution to 96x96 and increasing the number of Gabor wavelengths from 5 to 9 so that they spanned 2:32 pixels in 1/2 octave steps improved performance of the nonlinear AdaSVM to 93.3% correct. As the resolution goes up, the speed benefit of AdaSVM's becomes even more apparent. At the

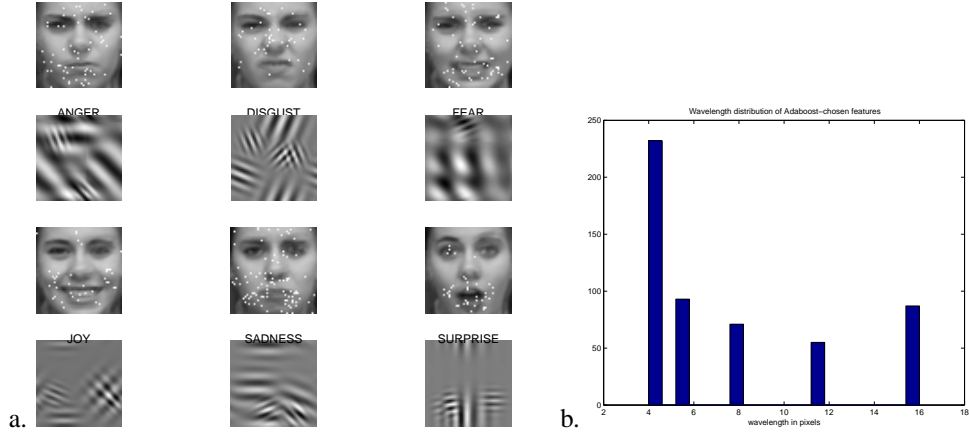

a.

b.

Figure 3: a. Gabors selected by Adaboost for each expression. White dots indicate locations of all selected Gabors. Below each expression is a linear combination of the real part of the first 5 Adaboost features selected for that expression. Faces shown are a mean of 10 individuals. b. Wavelength distribution of features selected by Adaboost.

higher resolution, the full Gabor representation increased by a factor of 7, whereas the number of Gabors selected by Adaboost only increased by a factor of 1.75.

Performance of the system was also evaluated on a second publicly available dataset, Pictures of Facial Affect[1]. We obtained 97% accuracy for generalization to novel subjects, trained by leave-one-subject-out cross-validation. This is about 10 percentage points higher than the best previously reported results on this dataset [9, 8].

An emergent property was that the outputs of the classifier change smoothly as a function of time, providing a potentially valuable representation to code facial expression dynamics in a fully automatic and unobtrusive manner. (See Figure 5.) In the next section, we apply this system to assessing spontaneous facial expressions in the field.

|  | Leave-group-out | | Leave-subject-out | |
|  | Adaboost | SVM | SVM | AdaSVM |
|---|---|---|---|---|
| Linear | 85.0 | 84.8 | 86.2 | 88.8 |
| RBF |  | 86.9 | 88.0 | 90.7 |

Table 1: Performance of Adaboost,SVM's and AdaSVM's (48x48 images).

|  | SVM | | Adaboost | AdaSVM | |
|  | Lin | RBF |  | Lin | RBF |
|---|---|---|---|---|---|
| Time t | t | 90t | 0.01t | 0.01t | 0.0125t |
| Time $t'$ | t | 90t | 0.16t | 0.16t | 0.2t |
| Memory | m | 90m | 3m | 3m | 3.3m |

Table 2: Processing time and memory considerations. Time $t'$ includes the extra time to calculate the outputs of the 538 Gabors in pixel space for Adaboost and AdaSVM, rather than the full FFT employed by the SVM's.

## 6   Deployment and evaluation: Automatic Evaluation of Human-Robot Interaction

We are currently evaluating the system as a tool for automatically measuring the quality of human-robot social interaction. This test involves recognition of spontaneous facial expressions in the continuous video stream during unconstrained interaction with RoboVie, a social robot under development at ATR and the University of Osaka [5]. This study was conducted at ATR in Kyoto, Japan. 14 participants, male and female, were instructed to interact with RoboVie for 5 minutes. Their facial expressions were recorded via 4 video cameras. The study was followed by a questionnaire in which the participants were asked to evaluate different aspects of their interaction with RoboVie.

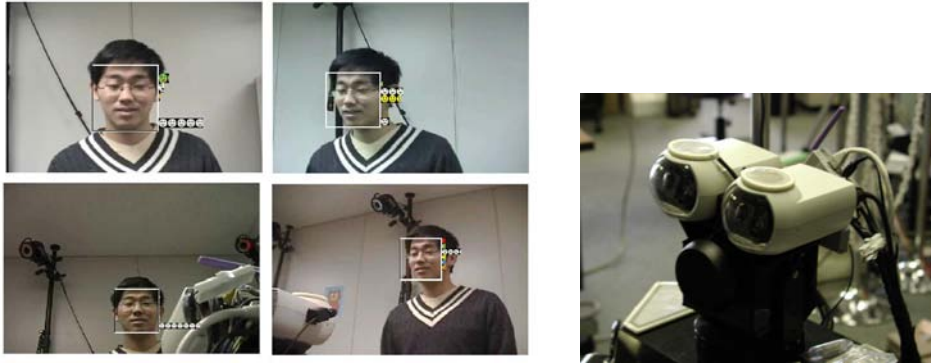

Figure 4: Human response during interaction with the RoboVie robot at ATR is measured by automatic expression analysis.

Faces were automatically detected and facial expressions classified in the continuous video streams of each of the four cameras. With the multi-camera paradigm, one or more cameras often provides a better view than the others. When the face is rotated, partially occluded, or misaligned, the expression classification is less reliable. A confidence measure from the face detection step consisted of the final unthresholded output of the cascade passed through a softmax transform over the four cameras. This measure indicated how much like a frontal face the system determined the selected window from each camera to be.

We compared the system's expression labels with a form of ground truth from human judgment. Four naive human observers were presented with the videos of each subject at 1/3 speed. The observers indicated the amount of happiness shown by the subject in each video by turning a dial.

The outputs of the four cameras were integrated by training a linear regression on 32 numbers, the continuous outputs of the seven emotion classifiers (the margin) plus the confidence measure from the face detector for each of the four cameras, to predict the human facial expression judgments. Figure 5 compares the human judgments with the automated system. Preliminary results are promising. The automated system predicted the human expression judgments with a correlation coefficient of 0.87, which was within the agreement range of the four human observers.[1]

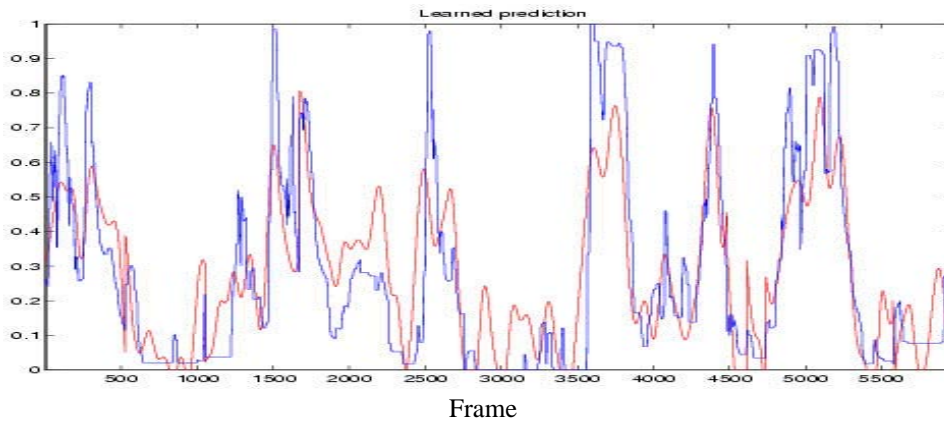

Frame

Figure 5: Human labels (blue/dark) compared to automated system labels (red/light) for 'joy' (one subject, one observer).

## 7 Conclusions

Computer animated agents and robots bring a social dimension to human computer interaction and force us to think in new ways about how computers could be used in daily life. Social robots and agents designed to recognize facial expression might provide a much more interesting and engaging social interaction, which can benefit applications from automated tutors to entertainment robots. Face to face communication is a real-time process operating at a time scale of less than a second. The level of uncertainty at this time scale is considerable, making it necessary for humans and machines to rely on sensory rich perceptual primitives rather than slow symbolic inference processes. In this paper we present progress on one such perceptual primitive: Real time recognition of facial expressions.

Our results suggest that user independent fully automatic real time coding of basic expressions is an achievable goal with present computer power, at least for applications in which frontal views or multiple cameras can be assumed. Good performance results were obtained for directly processing the output of an automatic face detector without the need for explicit detection and registration of facial features. A novel classification technique was presented that combines feature selection based on Adaboost with feature integration based on support vector machines. The AdaSVM's outperformed Adaboost and SVM's alone, and gave a considerable advantage in speed over SVM's. Strong performance results, 93% and 97% accuracy for generalization to novel subjects, were presented for two publicly available datasets of facial expressions collected by experimental psychologists expert in facial expressions.

We introduced a technique for automatically evaluating the quality of human-robot interaction based on the analysis of facial expressions. This test involved recognition of spontaneous facial expressions in the continuous video stream during unconstrained behavior. The system predicted human judgements of joy with a correlation of 0.87.

Within the past decade, significant advances in machine learning and machine perception open up the possibility of automatic analysis of facial expressions. Automated systems will have a tremendous impact on basic research by making facial expression measurement more accessible as a behavioral measure, and by providing data on the dynamics of facial behavior at a resolution that was previously unavailable. Such systems will also lay the foundations for computers that can understand this critical aspect of human communication. Computer systems with this capability have a wide range of applications in basic and applied research areas, including man-machine communication, security, law enforcement, psychiatry, education, and telecommunications.

## Acknowledgments

Support for this project was provided by ONR N00014-02-1-0616, NSF-ITR IIS-0220141 and IIS-0086107, DCI contract No.2000-I-058500-000, and California Digital Media Innovation Program DiMI 01-10130, and the MIND Institute. This research was supported in part by the Telecommunications Advancement Organization of Japan.

## Footnotes

[1]These are results from one subject. Test results based on 14 subjects will be available in one week. We are also comparing facial expression measurements by both human and computer to the self-report questionnaires.

## References

[1] P. Ekman and W. Friesen. Pictures of facial affect. Photographs, 1976. Available from Human Interaction Laboratory, UCSF, HIL-0984, San Francisco, CA 94143.

[2] I. Fasel and J. R. Movellan. Comparison of neurally inspired face detection algorithms. In *Proceedings of the international conference on artificial neural networks (ICANN 2002)*. UAM, 2002.

[3] Yoav Freund and Robert E. Schapire. Experiments with a new boosting algorithm. In *Proc. 13th International Conference on Machine Learning*, pages 148–146. Morgan Kaufmann, 1996.

[4] J Friedman, T Hastie, and R Tibshirani. Additive logistic regression: A statistical view of boosting. *ANNALS OF STATISTICS*, 28(2):337–374, 2000.

[5] H. Ishiguro, T. Ono, M. Imai, T. Maeda, and T. Kandaand R. Nakatsu. Robovie: an interactive humanoid robot. 28(6):498–503, 2001.

[6] T. Kanade, J.F. Cohn, and Y. Tian. Comprehensive database for facial expression analysis. In *Proceedings of the fourth IEEE International conference on automatic face and gesture recognition (FG'00)*, pages 46–53, Grenoble, France, 2000.

[7] M. Lades, J. Vorbrüggen, J. Buhmann, J. Lange, W. Konen, C. von der Malsburg, and R. Würtz. Distortion invariant object recognition in the dynamic link architecture. *IEEE Transactions on Computers*, 42(3):300–311, 1993.

[8] M. Lyons, J. Budynek, A. Plante, and S. Akamatsu. Classifying facial attributes using a 2-d gabor wavelet representation and discriminant analysis. In *Proceedings of the 4th international conference on automatic face and gesture recognition*, pages 202–207, 2000.

[9] C. Padgett and G. Cottrell. Representing face images for emotion classification. In M. Mozer, M. Jordan, and T. Petsche, editors, *Advances in Neural Information Processing Systems*, volume 9, Cambridge, MA, 1997. MIT Press.

[10] H. Rowley, S. Baluja, and T. Kanade. Neural network-based face detection. *IEEE Trans. on Pattern Analysis and Machine Intelligence*, 1(20):23–28, 1998.

[11] H. Schneiderman and T. Kanade. Probabilistic modeling of local appearance and spatial relationships for object recognition. In *Proc. IEEE Intl. Conf. on Computer Vision and Pattern Recognition*, pages 45–51, 1998.

[12] Kah Kay Sung and Tomaso Poggio. Example based learning for view-based human face detection. Technical Report AIM-1521, 1994.

[13] Paul Viola and Michael Jones. Robust real-time object detection. Technical Report CRL 20001/01, Cambridge ResearchLaboratory, 2001.
